# Manifold Embeddings for Model-Based Reinforcement Learning under Partial Observability

**Keith Bush**
School of Computer Science
McGill University
Montreal, Canada
kbush@cs.mcgill.ca

**Joelle Pineau**
School of Computer Science
McGill University
Montreal, Canada
jpineau@cs.mcgill.ca

## Abstract

Interesting real-world datasets often exhibit nonlinear, noisy, continuous-valued states that are unexplorable, are poorly described by first principles, and are only partially observable. If partial observability can be overcome, these constraints suggest the use of model-based reinforcement learning. We experiment with manifold embeddings to reconstruct the observable state-space in the context of off-line, model-based reinforcement learning. We demonstrate that the embedding of a system can change as a result of learning, and we argue that the best performing embeddings well-represent the dynamics of both the uncontrolled and adaptively controlled system. We apply this approach to learn a neurostimulation policy that suppresses epileptic seizures on animal brain slices.

## 1   Introduction

The accessibility of large quantities of off-line discrete-time dynamic data—state-action sequences drawn from real-world domains—represents an untapped opportunity for widespread adoption of reinforcement learning. By real-world we imply domains that are characterized by continuous state, noise, and partial observability. Barriers to making use of this data include: 1) goals (rewards) are not well-defined, 2) exploration is expensive (or not permissible), and 3) the data does not preserve the Markov property. If we assume that the reward function is part of the problem description, then to learn from this data we must ensure the Markov property is preserved before we approximate the optimal policy with respect to the reward function in a model-free or model-based way.

For many domains, particularly those governed by differential equations, we may leverage the inductive bias of locality during function approximation to satisfy the Markov property. When applied to model-free reinforcement learning, function approximation typically assumes that the value function maps nearby states to similar expectations of future reward. As part of model-based reinforcement learning, function approximation additionally assumes that similar actions map to nearby future states from nearby current states [10]. Impressive performance and scalability of local model-based approaches [1, 2] and global model-free approaches [6, 17] have been achieved by exploiting the locality of dynamics in fully observable state-space representations of challenging real-world problems.

In partially observable systems, however, locality is not preserved without additional context. First principle models offer some guidance in defining local dynamics, but the existence of known first principles cannot always be assumed. Rather, we desire a general framework for reconstructing state-spaces of partially observable systems which guarantees the preservation of locality. Nonlinear dynamic analysis has long used manifold embeddings to reconstruct locally Euclidean state-spaces of unforced, partially observable systems [24, 18] and has identified ways of finding these embeddings non-parametrically [7, 12]. Dynamicists have also used embeddings as generative models of partially observable unforced systems [16] by numerically integrating over the resultant embedding.

Recent advances have extended the theory of manifold embeddings to encompass deterministically and stochastically forced systems [21, 22].

A natural next step is to apply these latest theoretical tools to reconstruct and control partially observable forced systems. We do this by first identifying an appropriate embedding for the system of interest and then leveraging the resultant locality to perform reinforcement learning in a model-based way. We believe it may be more practical to address reinforcement learning under partial observability in a model-based way because it facilitates reasoning about domain knowledge and off-line validation of the embedding parameters.

The primary contribution of this paper is to formally combine and empirically evaluate these existing, but not well-known, methods by incorporating them in off-line, model-based reinforcement learning of two domains. First, we study the use of embeddings to learn control policies in a partially observable variant of the well-known Mountain Car domain. Second, we demonstrate the embedding-driven, model-based technique to learn an effective and efficient neurostimulation policy for the treatment of epilepsy. The neurostimulation example is important because it resides among the hardest classes of learning domain—a continuous-valued state-space that is nonlinear, partially observable, prohibitively expensive to explore, noisy, and governed by dynamics that are currently not well-described by mathematical models drawn from first principles.

## 2 Methods

In this section we combine reinforcement learning, partial observability, and manifold embeddings into a single mathematical formalism. We then describe non-parametric means of identifying the manifold embedding of a system and how the resultant embedding may be used as a local model.

### 2.1 Reinforcement Learning

Reinforcement learning (RL) is a class of problems in which an agent learns an optimal solution to a multi-step decision task by interacting with its environment [23]. Many RL algorithms exist, but we will focus on the $Q$-learning algorithm.

Consider an environment (i.e. forced system) having a state vector, $\mathbf{s} \in \mathbb{R}^M$, which evolves according to a nonlinear differential equation but is discretized in time and integrated numerically according to the map, $f$. Consider an agent that interacts with the environment by selecting action, $a$, according to a policy function, $\pi$. Consider also that there exists a reward function, $g$, which informs the agent of the scalar *goodness* of taking an action with respect to the goal of some multi-step decision task. Thus, for each time, $t$,

$$
\begin{align}
a(t) &= \pi(\mathbf{s}(t)), \tag{1}\\
\mathbf{s}(t+1) &= f(\mathbf{s}(t), a(t)), \text{ and} \tag{2}\\
r(t+1) &= g(\mathbf{s}(t), a(t)). \tag{3}
\end{align}
$$

RL is the process of learning the optimal policy function, $\pi^*$, that maximizes the expected sum of future rewards, termed the optimal action-value function or $Q$-function, $Q^*$, such that,

$$Q^*(\mathbf{s}(t), a(t)) = r(t+1) + \gamma \max_a Q^*(\mathbf{s}(t+1), a), \tag{4}$$

where $\gamma$ is the discount factor on $[0, 1)$. Equation 4 assumes that $Q^*$ is known. Without *a priori* knowledge of $Q^*$ an approximation, $Q$, must be constructed iteratively. Assume the current $Q$-function estimate, $Q$, of the optimal, $Q^*$, contains error, $\delta$,

$$\delta(t) = r(t+1) + \gamma \max_a Q(\mathbf{s}(t+1), a) - Q(\mathbf{s}(t), a(t)),$$

where $\delta(t)$ is termed the *temporal difference error* or TD-error. The TD-error can be used to improve the approximation of $Q$ by

$$Q(\mathbf{s}(t), a(t)) = Q(\mathbf{s}(t), a(t)) + \alpha \delta(t), \tag{5}$$

where $\alpha$ is the learning rate. By selecting action $a$ that maximizes the current estimate of $Q$, $Q$-learning specifies that over many applications of Equation 5, $Q$ approaches $Q^*$.

## 2.2 Manifold Embeddings for Reinforcement Learning Under Partial Observability

$Q$-learning relies on complete state observability to identify the optimal policy. Nonlinear dynamic systems theory provides a means of reconstructing complete state observability from incomplete state via the method of delayed embeddings, formalized by Takens' Theorem [24]. Here we present the key points of Takens' Theorem utilizing the notation of Huke [8] in a deterministically forced system.

Assume $\mathbf{s}$ is an $M$-dimensional, real-valued, bounded vector space and $a$ is a real-valued action input to the environment. Assuming that the state update $f$ and the policy $\pi$ are deterministic functions, Equation 1 may be substituted into Equation 2 to compose a new function, $\phi$,

$$
\begin{aligned}
\mathbf{s}(t+1) &= f\left(\mathbf{s}(t), \pi(\mathbf{s}(t))\right), \\
&= \phi(\mathbf{s}(t)),
\end{aligned}
\tag{6}
$$

which specifies the discrete time evolution of the agent acting on the environment. If $\phi$ is a smooth map $\phi : \mathbb{R}^M \to \mathbb{R}^M$ and this system is observed via function, $y$, such that

$$
\tilde{s}(t) = y(\mathbf{s}(t)),
\tag{7}
$$

where $y : \mathbb{R}^M \to \mathbb{R}$, then if $\phi$ is invertible, $\phi^{-1}$ exists, and $\phi$, $\phi^{-1}$, and $y$ are continuously differentiable we may apply Takens' Theorem [24] to reconstruct the complete state-space of the observed system. Thus, for each $\tilde{s}(t)$, we can construct a vector $\mathbf{s}_E(t)$,

$$
\mathbf{s}_E(t) = [\tilde{s}(t), \tilde{s}(t-1), ..., \tilde{s}(t-(E-1))], \ \mathrm{E} > 2\mathrm{M},
\tag{8}
$$

such that $\mathbf{s}_E$ lies on a subset of $\mathbb{R}^E$ which is an embedding of $\mathbf{s}$. Because embeddings preserve the connectivity of the original vector-space, in the context of RL the mapping $\psi$,

$$
\mathbf{s}_E(t+1) = \psi(\mathbf{s}_E(t)),
\tag{9}
$$

may be substituted for $f$ (Eqn. 6) and vectors $\mathbf{s}_E(t)$ may be substituted for corresponding vectors $\mathbf{s}(t)$ in Equations 1–5 without loss of generality.

## 2.3 Non-parametric Identification of Manifold Embeddings

Takens' Theorem does not define how to compute the embedding dimension of arbitrary sequences of observations, nor does it provide a test to determine if the theorem is applicable. In general. the intrinsic dimension, $M$, of a system is unknown. Finding high-quality embedding parameters of challenging domains, such as chaotic and noise-corrupted nonlinear signals, occupy much of the fields of subspace identification and nonlinear dynamic analysis. Numerous methods of note exist, drawn from both disciplines. We employ a spectral approach [7]. This method, premised by the singular value decomposition (SVD), is non-parametric, computationally efficient, and robust to additive noise—all of which are useful in practical application. As will be seen in succeeding sections, this method finds embeddings which are both accurate in theoretical tests and useful in practice.

We summarize the spectral parameter selection algorithm as follows. Given a sequence of state observations $\tilde{\mathbf{s}}$ of length $\tilde{S}$, we choose a *sufficiently large* fixed embedding dimension, $\hat{E}$. Sufficiently large refers to a cardinality of dimension which is certain to be greater than twice the dimension in which the actual state-space resides. For each embedding window size, $\hat{T}_{\min} \in \{\hat{E}, ..., \tilde{S}\}$, we: 1) define a matrix $\mathbf{S}_{\hat{E}}$ having row vectors, $\mathbf{s}_{\hat{E}}(t), t \in \{\hat{T}_{\min}, ..., \tilde{S}\}$, constructed according to the rule,

$$
\mathbf{s}_{\hat{E}}(t) = [\tilde{s}(t), \tilde{s}(t-\tau), ..., \tilde{s}(t-(\hat{E}-1)\tau)],
\tag{10}
$$

where $\tau = \hat{T}_{\min}/(\hat{E}-1)$, 2) compute the SVD of the matrix $\mathbf{S}_{\hat{E}}$, and 3) record the vector of singular values, $\sigma(\hat{T}_{\min})$. Embedding parameters of $\tilde{\mathbf{s}}$ are found by analysis of the second singular values, $\sigma_2(\hat{T}_{\min})$, $\hat{T}_{\min} \in \{\hat{E}, ..., \tilde{S}\}$. The $\hat{T}_{\min}$ value of the first local maxima of this sequence is the approximate embedding window, $T_{\min}$, of $\tilde{\mathbf{s}}$. The approximate embedding dimension, $E$, is the number of *non-trivial* singular values of $\sigma(T_{\min})$ where we define non-trivial as a value greater than the long-term trend of $\sigma_{\hat{E}}$ with respect to $\hat{T}_{\min}$. Embedding $\tilde{\mathbf{s}}$ according to Equation 10 via parameters $E$ and $T_{\min}$ yields the matrix $\mathbf{S}_E$ of row vectors, $\mathbf{s}_E(t), t \in \{T_{\min}, ..., \tilde{S}\}$.

## 2.4 Generative Local Models from Embeddings

The preservation of locality and dynamics afforded by the embedding allows an approximation of the underlying dynamic system. To model this space we assume that the derivative of the Voronoi region surrounding each embedded point is well-approximated by the derivative at the point itself, a *nearest-neighbors* derivative [16]. Using this, we simulate trajectories as iterative numerical integration of the local state and gradient. We define the model and integration process formally.

Consider a dataset $\mathcal{D}$ as a set of temporally aligned sequences of state observations $\tilde{s}(t)$, action observations $a(t)$, and reward observations $r(t)$, $t \in \{1, ..., \tilde{S}\}$. Applying the spectral embedding method to $\mathcal{D}$ yields a sequence of vectors $\mathbf{s}_E(t)$ in $\mathbb{R}^E$ indexed by $t \in \{T_{\min}, ..., \tilde{S}\}$. A local model $\mathbf{M}$ of $\mathcal{D}$ is the set of 3-tuples, $\mathbf{m}(t) = \{\mathbf{s}_E(t), a(t), r(t)\}$, $t \in \{T_{\min}, ..., \tilde{S}\}$, as well as operations on these tuples, $\mathcal{A}(\mathbf{m}(t)) \equiv a(t)$, $\mathcal{S}(\mathbf{m}(t)) \equiv \mathbf{s}_E(t)$, $\mathcal{Z}(\mathbf{m}(t)) \equiv \mathbf{z}(t)$ where $\mathbf{z}(t) = [\mathbf{s}(t), a(t)]$, and $\mathcal{U}(\mathbf{M}, a) \equiv \mathbf{M}_a$ where $\mathbf{M}_a$ is the subset of tuples in $\mathbf{M}$ containing action $a$.

Consider a state vector $\mathbf{x}(i)$ in $\mathbb{R}^E$ indexed by simulation time, $i$. To numerically integrate this state we define the gradient according to our definition of locality, namely the nearest neighbor. This step is defined differently for models having discrete and continuous actions. The model's nearest neighbor of $\mathbf{x}(i)$ when taking action $a(i)$ is defined in the case of a discrete set of actions, $A$, according to Equation 11 and in the continuous case it is defined by Equation 12,

$$\mathbf{m}(t_{\mathbf{x}(i)}) = \underset{\mathbf{m(t)} \in \mathcal{U}(\mathbf{M}, a(i))}{\arg\min} \|\mathcal{S}(\mathbf{m}(t)) - \mathbf{x}(i)\|, \, a \in A, \tag{11}$$

$$\mathbf{m}(t_{\mathbf{x}(i)}) = \underset{\mathbf{m(t)} \in \mathbf{M}}{\arg\min} \|\mathcal{Z}(\mathbf{m}(t)) - [\mathbf{x}(i), \omega a(i)]\|, \, a \in \mathbb{R}. \tag{12}$$

where $\omega$ is a scaling parameter on the action space. The model gradient and numerical integration are defined, respectively, as,

$$\nabla_{\mathbf{x}(i)} = \mathcal{S}(\mathbf{m}(t_{\mathbf{x}(i)} + 1)) - \mathcal{S}(\mathbf{m}(t_{\mathbf{x}(i)})) \text{ and} \tag{13}$$

$$\mathbf{x}(i+1) = \mathbf{x}(i) + \Delta i \left( \nabla_{\mathbf{x}(i)} + \eta \right), \tag{14}$$

where $\eta$ is a vector of noise and $\Delta i$ is the integration step-size. Applying Equations 11–14 iteratively simulates a trajectory of the underlying system, termed a *surrogate trajectory*. Surrogate trajectories are initialized from state $\mathbf{x}(0)$. Equation 14 assumes that dataset $\mathcal{D}$ contains noise. This noise biases the derivative estimate in $\mathbb{R}^E$, via the embedding rule (Eqn. 10). In practice, a small amount of additive noise facilitates generalization.

## 2.5 Summary of Approach

Our approach is to combine the practices of dynamic analysis and RL to construct useful policies in partially observable, real-world domains via off-line learning. Our meta-level approach is divided into two phases: the modeling phase and the learning phase.

We perform the modeling phase in steps: 1) record a partially observable system (and its rewards) under the control of a random policy or some other policy or set of policies that include observations of high reward value; 2) identify good candidate parameters for the embedding via the spectral embedding method; and 3) construct the embedding vectors and define the local model of the system.

During the learning phase, we identify the optimal policy on the local model with respect to the rewards, $\mathcal{R}(\mathbf{m}(t)) \equiv r(t)$, via batch $Q$-learning. In this work we consider strictly local function approximation of the model and $Q$-function, thus, we define the $Q$-function as a set of values, $Q$, indexed by the model elements, $Q(\mathbf{m}), \mathbf{m} \in \mathbf{M}$. For a state vector $\mathbf{x}(i)$ in $\mathbb{R}^E$ at simulation time $i$, and an associated action, $a(i)$, the reward and $Q$-value of this state can be indexed by either Equation 11 or 12, depending on whether the action is discrete or continuous. Note, our technique does not preclude the use of non-local function approximation, but here we assume a sufficient density of data exists to reconstruct the embedded state-space with minimal bias.

# 3 Case Study: Mountain Car

The Mountain Car problem is a second-order, nonlinear dynamic system with low-dimensional, continuous-valued state and action spaces. This domain is perhaps the most studied continuous-valued RL domain in the literature, but, surprisingly, there is little study of the problem in the case where the velocity component of state is unobserved. While not a real-world domain as imagined in the introduction, Mountain Car provides a familiar benchmark to evaluate our approach.

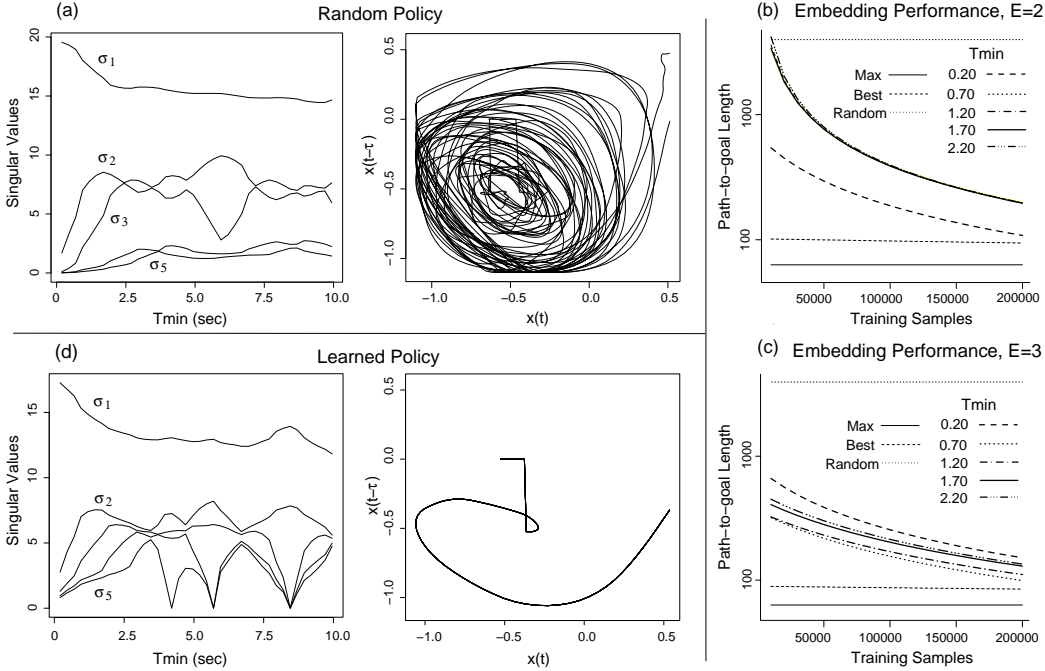

Figure 1: Learning experiments on Mountain Car under partial observability. (a) Embedding spectrum and accompanying trajectory ($E = 3$, $T_{\min} = 0.70$ sec.) under random policy. (b) Learning performance as a function of embedding parameters and quantity of training data. (c) Embedding spectrum and accompanying trajectory ($E = 3$, $T_{\min} = 0.70$ sec.) for the learned policy.

We use the Mountain Car dynamics and boundaries of Sutton and Barto [23]. We fix the initial state for all experiments (and resets) to be the lowest point of the mountain domain with zero velocity, which requires the longest path-to-goal in the optimal policy. Only the position element of the state is observable. During the modeling phase, we record this domain under a random control policy for 10,000 time-steps ($\Delta t = 0.05$ seconds), where the action is changed every $\Delta t = 0.20$ seconds. We then compute the spectral embedding of the observations ($T_{\min} = [0.20, 9.95]$ sec., $\Delta T_{\min} = 0.25$ sec., and $\hat{E} = 5$). The resulting spectrum is presented in Figure 1(a). We conclude that the embedding of Mountain Car under the random policy requires dimension $E = 3$ with a maximum embedding window of $T_{\min} = 1.70$ seconds.

To evaluate learning phase outcomes with respect to modeling phase outcomes, we perform an experiment where we model the randomly collected observations using embedding parameters drawn from the product of the sets $T_{\min} = \{0.20, 0.70, 1.20, 1.70, 2.20\}$ seconds and $E = \{2, 3\}$. While we fix the size of the local model to 10,000 elements we vary the total amount of training samples observed from 10,000 to 200,000 at intervals of 10,000. We use batch $Q$-learning to identify the optimal policy in a model-based way—in Equation 5 the transition between state-action pair and the resulting state-reward pair is drawn from the model ($\eta = 0.001$). After learning converges, we execute the learned policy on the real system for 10,000 time-steps, recording the mean path-to-goal length over all goals reached. Each configuration is executed 30 times.

We summarize the results of these experiments by log-scale plots, Figures 1(b) and (c), for embeddings of dimension two and three, respectively. We compare learning performance against three measures: the maximum performing policy achievable given the dynamics of the system (path-to-goal = 63 steps), the best (99th percentile) learned policy for each quantity of training data for each embedding dimension, and the random policy. Learned performance is plotted as linear regression fits of the data.

Policy performance results of Figures 1(b) and (c) may be summarized by the following observations. Performance positively relates to the quantity of off-line training data for all embedding parameters. Except for the configuration ($E = 2$, $T_{\min} = 0.20$), influence of $T_{\min}$ on learning performance relative to $E$ is small. Learning performance of 3-dimensional embeddings dominate

all but the shortest 2-dimensional embeddings. These observations indicate that the parameters of the embedding ultimately determine the effectiveness of RL under partial observability. This is not surprising. What is surprising is that the best performing parameter configurations are linked to dynamic characteristics of the system under both a random policy *and* the learned policy.

To support this claim we collected 1,000 sample observations of the best policy ($E = 3$, $T_{\min} = 0.70$ sec., $N_{train} = 200,000$) during control of the real Mountain Car domain (path-to-goal = 79 steps). We computed and plotted the embedding spectrum and first two dimensions of the embedding in Figure 1(d). We compare these results to similar plots for the random policy in Figure 1(a). We observe that the spectrum of the learned system has shifted such that the optimal embedding parameters require a shorter embedding window, $T_{\min} = 0.70$–$1.20$ sec. and a lower embedding dimension $E = 2$ (i.e., $\sigma_3$ peaks at $T_{\min} = 0.70$–$1.20$ and $\sigma_3$ falls below the trend of $\sigma_5$ at this window length). We confirm this by observing the embedding directly, Figure 1(d). Unlike the random policy, which includes both an unstable spiral fixed point and limit cycle structure and requires a 3-dimensional embedding to preserve locality, the learned policy exhibits a 2-dimensional unstable spiral fixed point. Thus, the fixed-point structure (embedding structure) of the combined policy-environment system changes during learning.

To reinforce this claim, we consider the difference between a 2-dimensional and 3-dimensional embedding. An agent may learn to project into a 2-dimensional plane of the 3-dimensional space, thus decreasing its embedding dimension *if* the training data supports a 2-dimensional policy. We believe it is no accident that ($E = 3$, $T_{\min} = 0.70$) is the best performing configuration across all quantities of training data. This configuration can represent both 3-dimensional and 2-dimensional policies, depending on the amount of training data available. It can also select between 2-dimensional embeddings having window sizes of $T_{\min} = \{0.35, 0.70\}$ sec., depending on whether the second or third dimension is projected out. One resulting parameter configuration ($E = 2$, $T_{\min} = 0.35$) is near the optimal 2-dimensional configuration of Figure 1(b).

## 4 Case Study: Neurostimulation Treatment of Epilepsy

Epilepsy is a common neurological disorder which manifests itself, electrophysiologically, in the form of intermittent seizures—intense, synchronized firing of neural populations. Researchers now recognize seizures as artifacts of abnormal neural dynamics and rely heavily on the nonlinear dynamic systems analysis and control literature to understand and treat seizures [4]. Promising techniques have emerged from this union. For example, fixed frequency electrical stimulation of slices of the rat hippocampus under artificially induced epilepsy have been demonstrated to suppress the frequency, duration, or amplitude of seizures [9, 5]. Next generation epilepsy treatments, derived from machine learning, promise maximal seizure suppression via minimal electrical stimulation by adapting control policies to patients' unique neural dynamics. Barriers to constructing these treatments arise from a lack of first principles understanding of epilepsy. Without first principles, neuroscientists have only vague notions of what effective neurostimulation treatments should look like. Even if effective policies could be envisioned, exploration of the vast space of policy parameters is impractical without computational models.

Our specific control problem is defined as follows. Given labeled field potential recordings of brain slices under fixed-frequency electrical stimulation policies of 0.5, 1.0, and 2.0 Hz, as well as unstimulated control data, similar to the time-series depicted in Figure 2(a), we desire to learn a stimulation policy that suppresses seizures of a real, previously unseen, brain slice with an effective mean frequency (number of stimulations divided by the time the policy is active) of less than 1.0 Hz (1.0 Hz is currently known to be the most robust suppression policy for the brain slice model we use [9, 5]). As a further complication, on-line exploration is extremely expensive because the brain slices are experimentally viable for periods of less than 2 hours.

Again, we approach this problem as separate modeling and learning phases. We first compute the embedding spectrum of our dataset assuming $\hat{E} = 15$, presented in Figure 2(b). Using our knowledge of the interaction between embedding parameters and learning we select the embedding dimension $E = 3$ and embedding window $T_{\min} = 1.05$ seconds. Note, the strong maxima of $\sigma_2$ at $T_{\min} = 110$ seconds is the result of periodicity of seizures in our small training dataset. Periodicity of spontaneous seizure formation, however, varies substantially between slices. We select a shorter embedding window and rely on integration of the local model to unmask long-term dynamics.

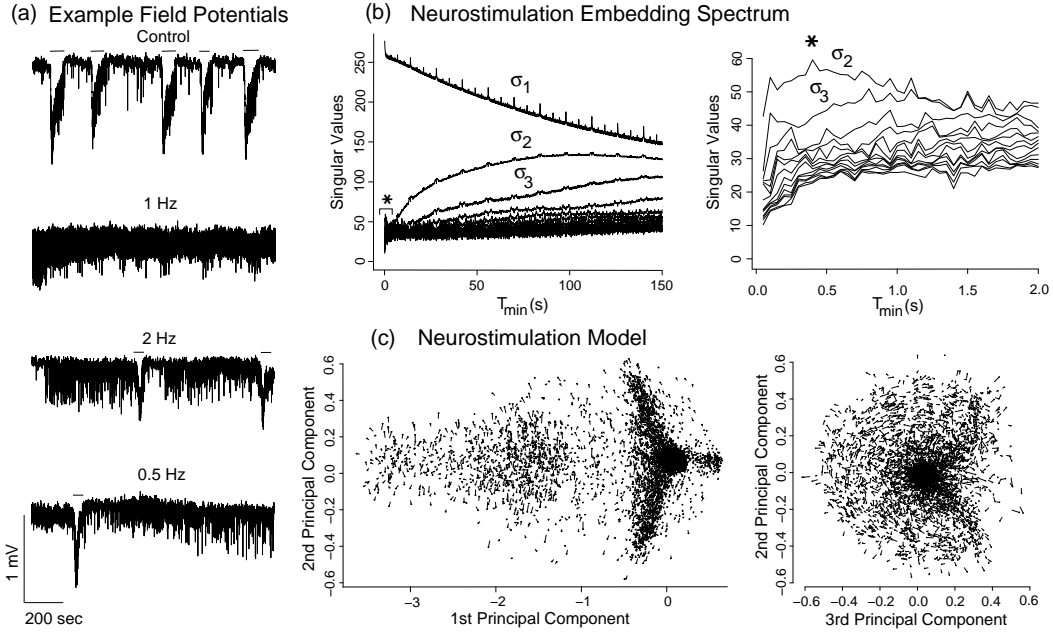

Figure 2: Graphical summary of the modeling phase of our adaptive neurostimulation study. (a) Sample observations from the fixed-frequency stimulation dataset. Seizures are labeled with horizontal lines. (b) The embedding spectrum of the fixed-frequency stimulation dataset. The large maximum of $\sigma_2$ at approximately 100 sec. is an artifact of the periodicity of seizures in the dataset. *Detail of the embedding spectrum for $T_{\min} = [0.05, 2.0]$ depicting a maximum of $\sigma_2$ at the time-scale of individual stimulation events. (c) The resultant neurostimulation model constructed from embedding the dataset with parameters ($E = 3$, $T_{\min} = 1.05$ sec.). Note, the model has been desampled $5\times$ in the plot.

In this complex domain we apply the spectral method differently than described in Section 2. Rather than building the model directly from the embedding ($E = 3$, $T_{\min} = 1.05$), we perform a change of basis on the embedding ($\hat{E} = 15$, $T_{\min} = 1.05$), using the first three columns of the right singular vectors, analogous to projecting onto the principal components. This embedding is plotted in Figure 2(c). Also, unlike the previous case study, we convert stimulation events in the training data from discrete frequencies to a continuous scale of time-elapsed-since-stimulation. This allows us to combine all of the data into a single state-action space and then simulate any arbitrary frequency. Based on exhaustive closed-loop simulations of fixed-frequency suppression efficacy across a spectrum of $[0.001, 2.0]$ Hz, we constrain the model's action set to discrete frequencies $a = \{2.0, 0.25\}$ Hz in the hopes of easing the learning problem. We then perform batch $Q$-learning over the model ($\Delta t = 0.05$, $\omega = 0.1$, and $\eta = 0.00001$), using discount factor $\gamma = 0.9$. We structure the reward function to penalize each electrical stimulation by $-1$ and each visited seizure state by $-20$.

Without stimulation, seizure states comprise 25.6% of simulation states. Under a 1.0 Hz fixed-frequency policy, stimulation events comprise 5.0% and seizures comprise 6.8% of the simulation states. The policy learned by the agent also reduces the percent of seizure states to 5.2% of simulation states while stimulating only 3.1% of the time (effective frequency equals 0.62 Hz). In simulation, therefore, the learned policy achieves the goal.

We then deployed the learned policy on real brain slices to test on-line seizure suppression performance. The policy was tested over four trials on two unique brain slices extracted from the same animal. The effective frequencies of these four trials were $\{0.65, 0.64, 0.66, 0.65\}$ Hz. In all trials seizures were effectively suppressed after a short transient period, during which the policy and slice achieved equilibrium. (Note: seizures occurring at the onset of stimulation are common artifacts of neurostimulation). Figure 3 displays two of these trials spaced over four sequential phases: (a) a control (no stimulation) phase used to determine baseline seizure activity, (b) a learned policy trial lasting 1,860 seconds, (c) a recovery phase to ensure slice viability after stimulation and to recompute baseline seizure activity, and (d) a learned policy trial lasting 2,130 seconds.

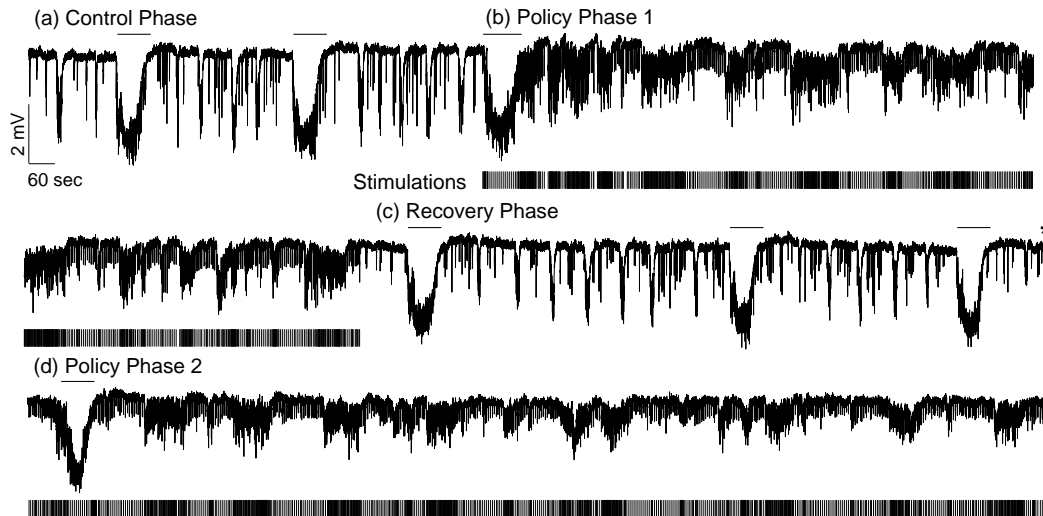

Figure 3: Field potential trace of a real seizure suppression experiment using a policy learned from simulation. Seizures are labeled as horizontal lines above the traces. Stimulation events are marked by vertical bars below the traces. (a) A control phase used to determine baseline seizure activity. (b) The initial application of the learned policy. (c) A recovery phase to ensure slice viability after stimulation and recompute baseline seizure activity. (d) The second application of the learned policy. *10 minutes of trace are omitted while the algorithm was reset.

## 5   Discussion and Related Work

The RL community has long studied low-dimensional representations to capture complex domains. Approaches for efficient function approximation, basis function construction, and discovery of embeddings has been the topic of significant investigations [3, 11, 20, 15, 13]. Most of this work has been limited to the fully observable (MDP) case and has not been extended to partially observable environments. The question of state space representation in partially observable domains was tackled under the POMDP framework [14] and recently in the PSR framework [19]. These methods address a similar problem but have been limited primarily to discrete action and observation spaces. The PSR framework was extended to continuous (nonlinear) domains [25]. This method is significantly different from our work, both in terms of the class of representations it considers and in the criteria used to select the appropriate representation. Furthermore, it has not yet been applied to real-world domains. An empirical comparison with our approach is left for future consideration.

The contribution of our work is to integrate embeddings with model-based RL to solve real-world problems. We do this by leveraging locality preserving qualities of embeddings to construct dynamic models of the system to be controlled. While not improving the quality of off-line learning that is possible, these models permit embedding validation and reasoning over the domain, either to constrain the learning problem or to anticipate the effects of the learned policy on the dynamics of the controlled system. To demonstrate our approach, we applied it to learn a neurostimulation treatment of epilepsy, a challenging real-world domain. We showed that the policy learned off-line from an embedding-based, local model can be successfully transferred on-line. This is a promising step toward widespread application of RL in real-world domains. Looking to the future, we anticipate the ability to adjust the embedding *a priori* using a non-parametric policy gradient approach over the local model. An empirical investigation into the benefits of this extension are also left for future consideration.

**Acknowledgments**

The authors thank Dr. Gabriella Panuccio and Dr. Massimo Avoli of the Montreal Neurological Institute for generating the time-series described in Section 4. The authors also thank Arthur Guez, Robert Vincent, Jordan Frank, and Mahdi Milani Fard for valuable comments and suggestions. The authors gratefully acknowledge financial support by the Natural Sciences and Engineering Research Council of Canada and the Canadian Institutes of Health Research.

# References

[1] Christopher G. Atkeson, Andrew W. Moore, and Stefan Schaal. Locally weighted learning for control. *Artificial Intelligence Review*, 11:75–113, 1997.

[2] Christopher G. Atkeson and Jun Morimoto. Nonparametric representation of policies and value functions: A trajectory-based approach. In *Advances in Neural Information Processing*, 2003.

[3] M. Bowling, A. Ghodsi, and D. Wilkinson. Action respecting embedding. In *Proceedings of ICML*, 2005.

[4] F. Lopes da Silva, W. Blanes, S. Kalitzin, J. Parra, P. Suffczynski, and D. Velis. Dynamical diseases of brain systems: Different routes to epileptic seizures. *IEEE Transactions on Biomedical Engineering*, 50(5):540–548, 2003.

[5] G. D'Arcangelo, G. Panuccio, V. Tancredi, and M. Avoli. Repetitive low-frequency stimulation reduces epileptiform synchronization in limbic neuronal networks. *Neurobiology of Disease*, 19:119–128, 2005.

[6] Damien Ernst, Pierre Guerts, and Louis Wehenkel. Tree-based batch mode reinforcement learning. *Journal of Machine Learning Research*, 6:503–556, 2005.

[7] A. Galka. *Topics in Nonlinear Time Series Analysis: with implications for EEG Analysis*. World Scientific, 2000.

[8] J.P. Huke. Embedding nonlinear dynamical systems: A guide to Takens' Theorem. Technical report, Manchester Institute for Mathematical Sciences, University of Manchester, March, 2006.

[9] K. Jerger and S. Schiff. Periodic pacing and in vitro epileptic focus. *Journal of Neurophysiology*, 73(2):876–879, 1995.

[10] Nicholas K. Jong and Peter Stone. Model-based function approximation in reinforcement learning. In *Proceedings of AAMAS*, 2007.

[11] P.W. Keller, S. Mannor, and D. Precup. Automatic basis function construction for approximate dynamic programming and reinforcement learning. In *Proceedings of ICML*, 2006.

[12] M. Kennel and H. Abarbanel. False neighbors and false strands: A reliable minimum embedding dimension algorithm. *Physical Review E*, 66:026209, 2002.

[13] S. Mahadevan and M. Maggioni. Proto-value functions: A Laplacian framework for learning representation and control in Markov decision processes. *Journal of Machine Learning Research*, 8:2169–2231, 2007.

[14] A. K. McCallum. *Reinforcement Learning with Selective Perception and Hidden State*. PhD thesis, University of Rochester, 1996.

[15] R. Munos and A. Moore. Variable resolution discretization in optimal control. *Machine Learning*, 49:291–323, 2002.

[16] U. Parlitz and C. Merkwirth. Prediction of spatiotemporal time series based on reconstructed local states. *Physical Review Letters*, 84(9):1890–1893, 2000.

[17] Jan Peters, Sethu Vijayakumar, and Stefan Schaal. Natural actor-critic. In *Proceedings of ECML*, 2005.

[18] Tim Sauer, James A. Yorke, and Martin Casdagli. Embedology. *Journal of Statistical Physics*, 65:3/4:579–616, 1991.

[19] S. Singh, M. L. Littman, N. K. Jong, D. Pardoe, and P. Stone. Learning predictive state representations. In *Proceedings of ICML*, 2003.

[20] W. Smart. Explicit manifold representations for value-functions in reinforcement learning. In *Proceedings of ISAIM*, 2004.

[21] J. Stark. Delay embeddings for forced systems. I. Deterministic forcing. *Journal of Nonlinear Science*, 9:255–332, 1999.

[22] J. Stark, D.S. Broomhead, M.E. Davies, and J. Huke. Delay embeddings for forced systems. II. Stochastic forcing. *Journal of Nonlinear Science*, 13:519–577, 2003.

[23] R. Sutton and A. Barto. *Reinforcement learning: An introduction*. The MIT Press, Cambridge, MA, 1998.

[24] F. Takens. Detecting strange attractors in turbulence. In D. A. Rand & L. S. Young, editor, *Dynamical Systems and Turbulence*, volume 898, pages 366–381. Warwick, 1980.

[25] D. Wingate and S. Singh. On discovery and learning of models with predictive state representations of state for agents with continuous actions and observations. In *Proceedings of AAMAS*, 2007.

